# Beating SGD: Learning SVMs in Sublinear Time

**Elad Hazan**     **Tomer Koren**
Technion, Israel Institute of Technology
Haifa, Israel 32000
{ehazan@ie,tomerk@cs}.technion.ac.il

**Nathan Srebro**
Toyota Technological Institute
Chicago, Illinois 60637
nati@ttic.edu

## Abstract

We present an optimization approach for linear SVMs based on a stochastic primal-dual approach, where the primal step is akin to an importance-weighted SGD, and the dual step is a stochastic update on the importance weights. This yields an optimization method with a sublinear dependence on the training set size, and the first method for learning linear SVMs with runtime less then the size of the training set required for learning!

## 1   Introduction

Stochastic approximation (online) approaches, such as stochastic gradient descent and stochastic dual averaging, have become the optimization method of choice for many learning problems, including linear SVMs. This is not surprising, since such methods yield optimal generalization guarantees with only a single pass over the data. They therefore in a sense have optimal, unbeatable runtime: from a learning (generalization) point of view, in a "data laden" setting [2, 13], the runtime to get to a desired generalization goal is the same as the size of the data set required to do so. Their runtime is therefore equal (up to a small constant factor) to the runtime required to just *read* the data.

In this paper we show, for the first time, how to beat this unbeatable runtime, and present a method that, in a certain relevant regime of high dimensionality, relatively low noise and accuracy proportional to the noise level, learns in runtime *less* then the size of the minimal training set size required for generalization. The key here, is that unlike online methods that consider an entire training vector at each iteration, our method accesses single features (coordinates) of training vectors. Our computational model is thus that of random access to a desired coordinate of a desired training vector (as is standard for sublinear time algorithms), and our main computational cost are these feature accesses.

Our method can also be understood in the framework of "budgeted learning" [5] where the cost is explicitly the cost of observing features (but unlike, e.g. [8], we do not have differential costs for different features), and gives the first non-trivial guarantee in this setting (i.e. first theoretical guarantee on the number of feature accesses that is less then simply observing entire feature vectors).

We emphasize that our method is *not* online in nature, and we *do* require repeated access to training examples, but the resulting runtime (as well as the overall number of features accessed) is *less* (in some regimes) then for any online algorithms that considers entire training vectors. Also, unlike recent work by Cesa-Bianchi et al. [3], we are not constrained to only a few features from every vector, and can ask for however many we need (with the aim of minimizing the overall runtime, and thus the overall number of feature accesses), and so we obtain an overall number of feature accesses which is better then with SGD, unlike Cesa-Bianchi et al., which aim at not being too much worse then full-information SGD.

As discussed in Section 3, our method is a primal-dual method, where both the primal and dual steps are stochastic. The primal steps can be viewed as importance-weighted stochastic gradient descent, and the dual step as a stochastic update on the importance weighting, informed by the current primal solution. This approach builds on the work of [4] that presented a sublinear time algorithm for approximating the margin of a linearly separable data set. Here, we extend that work to the more rel-

evant noisy (non-separable) setting, and show how it can be applied to a learning problem, yielding generalization runtime better then SGD. The extension to the non-separable setting is not straight-forward and requires re-writing the SVM objective, and applying additional relaxation techniques borrowed from [10].

## 2 The SVM Optimization Problem

We consider training a linear binary SVM based on a training set of $n$ labeled points $\{\mathbf{x}_i, y_i\}_{i=1\ldots n}$, $\mathbf{x}_i \in \mathbb{R}^d$, $y_i \in \{\pm 1\}$, with the data normalized such that $\|\mathbf{x}_i\| \leq 1$. A predictor is specified by $\mathbf{w} \in \mathbb{R}^d$ and a bias $b \in \mathbb{R}$. In training, we wish to minimize the empirical error, measured in terms of the average hinge loss $\hat{R}_{\text{hinge}}(\mathbf{w}, b) = \frac{1}{n} \sum_{i=1}^n [1 - y(\langle \mathbf{w}, \mathbf{x}_i \rangle + b)]_+$ , and the norm of $\mathbf{w}$. Since we do not typically know a-priori how to balance the norm with the error, this is best described as an unconstrained bi-criteria optimization problem:

$$\min_{\mathbf{w} \in \mathbb{R}^d, b \in \mathbb{R}} \quad \|\mathbf{w}\| \quad , \quad \hat{R}_{\text{hinge}}(\mathbf{w}, b) \tag{1}$$

A common approach to finding Pareto optimal points of (1) is to scalarize the objective as:

$$\min_{\mathbf{w} \in \mathbb{R}^d, b \in \mathbb{R}} \hat{R}_{\text{hinge}}(\mathbf{w}, b) + \frac{\lambda}{2} \|\mathbf{w}\|^2 \tag{2}$$

where the multiplier $\lambda \geq 0$ controls the trade-off between the two objectives. However, in order to apply our framework, we need to consider a different parametrization of the Pareto optimal set (the "regularization path"): instead of minimizing a trade-off between the norm and the error, we maximize the margin (equivalent to minimizing the norm) subject to a constraint on the error. This allows us to write the objective (the margin) as a minimum over all training points—a form we will later exploit. Specifically, we introduce slack variables and consider the optimization problem:

$$\max_{\mathbf{w} \in \mathbb{R}^d,\, b \in \mathbb{R},\, 0 \leq \xi_i} \min_{i \in [n]} \quad y_i(\langle \mathbf{w}, \mathbf{x}_i \rangle + b) + \xi_i \qquad \text{s.t.} \quad \|\mathbf{w}\| \leq 1 \ \text{ and } \ \sum_{i=1}^n \xi_i \leq n\nu \tag{3}$$

where the parameter $\nu$ controls the trade-off between desiring a large margin (low norm) and small error (low slack), and parameterizes solutions along the regularization path. This is formalized by the following Lemma, which also gives guarantees for $\varepsilon$-sub-optimal solutions of (3):

**Lemma 2.1.** *For any $\mathbf{w} \neq 0$, $b \in \mathbb{R}$ consider problem (3) with $\nu = \hat{R}_{hinge}(\mathbf{w}, b)/\|\mathbf{w}\|$. Let $\mathbf{w}^\varepsilon, b^\varepsilon, \xi^\varepsilon$ be an $\varepsilon$-suboptimal solution to this problem with value $\gamma^\varepsilon$, and consider the rescaled solution $\tilde{\mathbf{w}} = \mathbf{w}^\varepsilon/\gamma^\varepsilon$, $\tilde{b} = b^\varepsilon/\gamma^\varepsilon$. Then:*

$$\|\tilde{\mathbf{w}}\| \leq \frac{1}{1 - \|\mathbf{w}\|\,\varepsilon} \|\mathbf{w}\|, \qquad \text{and} \qquad \hat{R}_{hinge}(\tilde{\mathbf{w}}) \leq \frac{1}{1 - \|\mathbf{w}\|\,\varepsilon} \hat{R}_{hinge}(\mathbf{w}).$$

That is, solving (3) exactly (to within $\varepsilon = 0$) yields Pareto optimal solutions of (1), and all such solutions (i.e. the entire regularization path) can be obtained by varying $\nu$. When (3) is only solved approximately, we obtain a Pareto sub-optimal point, as quantified by Lemma 2.1.

Before proceeding, we also note that any solution of (1) that classifies at least some positive and negative points within the desired margin must have $\|\mathbf{w}\| \geq 1$ and so in Lemma 2.1 we will only need to consider $0 \leq \nu \leq 1$. In terms of (3), this means that we could restrict $0 \leq \xi_i \leq 2$ without affecting the optimal solution.

## 3 Overview: Primal-Dual Algorithms and Our Approach

**The CHW framework**

The method of [4] applies to saddle-point problems of the form

$$\max_{\mathbf{z} \in \mathcal{K}} \min_{i \in [n]} c_i(\mathbf{z}). \tag{4}$$

where $c_i(\mathbf{z})$ are concave functions of $\mathbf{z}$ over some set $\mathcal{K} \subseteq \mathbb{R}^d$. The method is a stochastic primal-dual method, where the dual solution can be viewed as importance weighting over the $n$ terms $c_i(\mathbf{z})$. To better understand this view, consider the equivalent problem:

$$\max_{\mathbf{z} \in \mathcal{K}} \min_{p \in \Delta_n} \sum_{i=1}^{n} p_i c_i(\mathbf{z}) \tag{5}$$

where $\Delta_n = \{p \in \mathbb{R}^n \mid p_i \geq 0, \|p\|_1 = 1\}$ is the probability simplex. The method maintains and (stochastically) improves both a primal solution (in our case, a predictor $\mathbf{w} \in \mathbb{R}^d$) and a dual solution, which is a distribution $p$ over $[n]$. Roughly speaking, the distribution $p$ is used to focus in on the terms actually affecting the minimum. Each iteration of the method proceeds as follows:

1. **Stochastic primal update:**
   (a) A term $i \in [n]$ is chosen according to the distribution $p$, in time $O(n)$.
   (b) The primal variable $\mathbf{z}$ is updated according to the gradient of the $c_i(\mathbf{z})$, via an online low-regret update. This update is in fact a *Stochastic Gradient Descent* (SGD) step on the objective of (5), as explained in section 4. Since we use only a single term $c_i(\mathbf{z})$, this can be usually done in time $O(d)$.
2. **Stochastic dual update:**
   (a) We obtain a stochastic estimate of $c_i(\mathbf{z})$, for each $i \in [n]$. We would like to use an estimator that has a bounded variance, and can be computed in $O(1)$ time per term, i.e. in overall $O(n)$ time. When the $c_i$'s are linear functions, this can be achieved using a form of $\ell_2$-sampling for estimating an inner-product in $\mathbb{R}^d$.
   (b) The distribution $p$ is updated toward those terms with low estimated values of $c_i(\mathbf{z})$. This is accomplished using a variant of the *Multiplicative Updates* (MW) framework for online optimization over the simplex (see for example [1]), adapted to our case in which the updates are based on random variables with bounded variance. This can be done in time $O(n)$.

Evidently, the overall runtime per iteration is $O(n + d)$. In addition, the regret bounds on the updates of $\mathbf{z}$ and $p$ can be used to bound the number of iterations required to reach an $\varepsilon$-suboptimal solution. Hence, the CHW approach is particularly effective when this regret bound has a favorable dependence on $d$ and $n$. As we note below, this is not the case in our application, and we shall need some additional machinery to proceed.

**The PST framework**

The *Plotkin-Shmoys-Tardos* framework [10] is a deterministic primal-dual method, originally proposed for approximately solving certain types of linear programs known as "fractional packing and covering" problems. The same idea, however, applies also to saddle-point problems of the form (5).

In each iteration of this method, the primal variable $\mathbf{z}$ is updated by solving the "simple" optimization problem $\max_{\mathbf{z} \in \mathcal{K}} \sum_{i=1}^{n} p_i c_i(\mathbf{z})$ (where $p$ is now fixed), while the dual variable $p$ is again updated using a MW step (note that we do not use an estimation for $c_i(\mathbf{z})$ here, but rather the exact value). These iterations yield convergence to the optimum of (5), and the regret bound of the MW updates is used to derive a convergence rate guarantee.

Since each iteration of the framework relies on the entire set of functions $c_i$, it is reasonable to apply it only on relatively small-sized problems. Indeed, in our application we shall use this method for the update of the slack variables $\xi$ and the bias term $b$, for which the implied cost is only $O(n)$ time.

**Our hybrid approach**

The saddle-point formulation (3) of SVM from section 2 suggests that the SVM optimization problem can be efficiently approximated using primal-dual methods, and specifically using the CHW framework. Indeed, taking $\mathbf{z} = (\mathbf{w}, b, \xi)$ and $\mathcal{K} = \mathbb{B}_d \times [-1, 1] \times \Xi_\nu$ where $\mathbb{B}_d \subseteq \mathbb{R}^d$ is the Euclidean unit ball and $\Xi_\nu = \{\xi \in \mathbb{R}^n \mid \forall i\ 0 \leq \xi_i \leq 2,\ \|\xi\|_1 \leq \nu n\}$, we cast the problem into the form (4). However, as already pointed out, a naïve application of the CHW framework yields in this case a rather slow convergence rate. Informally speaking, this is because our set $\mathcal{K}$ is "too large" and thus the involved regret grows too quickly.

In this work, we propose a novel hybrid approach for tackling problems such as (3), that combines the ideas of the CHW and PST frameworks. Specifically, we suggest using a SGD-like low-regret

update for the variable $\mathbf{w}$, while updating the variables $\xi$ and $b$ via a PST-like step; the dual update of our method is similar to that of CHW. Consequently, our algorithm enjoys the benefits of both methods, each in its respective domain, and avoids the problem originating from the "size" of $\mathcal{K}$. We defer the detailed description of the method to the following section.

## 4   Algorithm and Analysis

In this section we present and analyze our algorithm, which we call SIMBA (stands for "*Sublinear IMportance-sampling Bi-stochastic Algorithm*"). The algorithm is a sublinear-time approximation algorithm for problem (3), which as shown in section 2 is a reformulation of the standard soft-margin SVM problem. For the simplicity of presentation, we omit the bias term for now (i.e., fix $b = 0$ in (3)) and later explain how adding such bias to our framework is almost immediate and does not affect the analysis. This allows us to ignore the labels $y_i$, by setting $\mathbf{x}_i \leftarrow -\mathbf{x}_i$ for any $i$ with $y_i = -1$.

Let us begin the presentation with some additional notation. To avoid confusion, we use the notation $v(i)$ to refer to the $i$'th coordinate of a vector $v$. We also use the shorthand $v^2$ to denote the vector for which $v^2(i) = (v(i))^2$ for all $i$. The $n$-vector whose entries are all $1$ is denoted as $\mathbf{1}_n$. Finally, we stack the training instances $\mathbf{x}_i$ as the *rows* of a matrix $\mathbf{X} \in \mathbb{R}^{n \times d}$, although we treat each $\mathbf{x}_i$ as a *column* vector.

---

**Algorithm 1** SVM-SIMBA

---

1: Input: $\varepsilon > 0$, $0 \leq \nu \leq 1$, and $\mathbf{X} \in \mathbb{R}^{n \times d}$ with $\mathbf{x}_i \in \mathbb{B}_d$ for $i \in [n]$.
2: Let $T \leftarrow 100^2 \varepsilon^{-2} \log n$, $\eta \leftarrow \sqrt{\log(n)/T}$ and $u_1 \leftarrow 0$, $q_1 \leftarrow \mathbf{1}_n$
3: **for** $t = 1$ to $T$ **do**
4:    Choose $i_t \leftarrow i$ with probability $p_t(i)$
5:    Let $u_t \leftarrow u_{t-1} + \mathbf{x}_{i_t}/\sqrt{2T}$, $\xi_t \leftarrow \arg\max_{\xi \in \Xi_\nu} (p_t^\top \xi)$
6:    $\mathbf{w}_t \leftarrow u_t / \max\{1, \|u_t\|\}$
7:    Choose $j_t \leftarrow j$ with probability $\mathbf{w}_t(j)^2/\|\mathbf{w}_t\|^2$.
8:    **for** $i = 1$ to $n$ **do**
9:        $\tilde{v}_t(i) \leftarrow \mathbf{x}_i(j_t)\|\mathbf{w}_t\|^2/\mathbf{w}_t(j_t) + \xi_t(i)$
10:       $v_t(i) \leftarrow \text{clip}(\tilde{v}_t(i), 1/\eta)$
11:       $q_{t+1}(i) \leftarrow q_t(i)(1 - \eta v_t(i) + \eta^2 v_t(i)^2)$
12:   **end for**
13:   $p_t \leftarrow q_t/\|q_t\|_1$
14: **end for**
15: **return** $\bar{\mathbf{w}} = \frac{1}{T}\sum_t \mathbf{w}_t$, $\bar{\xi} = \frac{1}{T}\sum_t \xi_t$

---

The pseudo-code of the SIMBA algorithm is given in figure 1. In the primal part (lines 4 through 6), the vector $u_t$ is updated by adding an instance $\mathbf{x}_i$, randomly chosen according to the distribution $p_t$. This is a version of SGD applied on the function $p_t^\top(\mathbf{X}\mathbf{w} + \xi_t)$, whose gradient with respect to $\mathbf{w}$ is $p_t^\top \mathbf{X}$; by the sampling procedure of $i_t$, the vector $\mathbf{x}_{i_t}$ is an unbiased estimator of this gradient. The vector $u_t$ is then projected onto the unit ball, to obtain $\mathbf{w}_t$. On the other hand, the primal variable $\xi_t$ is updated by a complete optimization of $p_t^\top \xi$ with respect to $\xi \in \Xi_\nu$. This is an instance of the PST framework, described in section 3. Note that, by the structure of $\Xi_\nu$, this update can be accomplished using a simple greedy algorithm that sets $\xi_t(i) = 2$ corresponding to the largest entries $p_t(i)$ of $p_t$, until a total mass of $\nu n$ is reached, and puts $\xi_t(i) = 0$ elsewhere; this can be implemented in $O(n)$ time using standard selection algorithms.

In the dual part (lines 7 through 13), the algorithm first updates the vector $q_t$ using the $j_t$ column of $\mathbf{X}$ and the value of $\mathbf{w}_t(j_t)$, where $j_t$ is randomly selected according to the distribution $\mathbf{w}_t^2/\|\mathbf{w}_t\|^2$. This is a variant of the MW framework (see definition 4.1 below) applied on the function $p^\top(\mathbf{X}\mathbf{w}_t + \xi_t)$; the vector $\tilde{v}$ serves as an estimator of $\mathbf{X}\mathbf{w}_t + \xi_t$, the gradient with respect to $p$. We note, however, that the algorithm uses a clipped version $v$ of the estimator $\tilde{v}$; see line 10, where we use the notation $\text{clip}(z, C) = \max(\min(z, C), -C)$ for $z, C \in \mathbb{R}$. This, in fact, makes $v$ a *biased* estimator of the gradient. As we show in the analysis, while the clipping operation is crucial to the stability of the algorithm, the resulting slight bias is not harmful.

Before stating the main theorem, we describe in detail the MW algorithm we use for the dual update.

**Definition 4.1** (Variance MW algorithm). *Consider a sequence of vectors $v_1, \ldots, v_T \in \mathbb{R}^n$ and a parameter $\eta > 0$. The Variance Multiplicative Weights (Variance MW) algorithm is as follows. Let $w_1 \leftarrow \mathbf{1}_n$, and for $t \geq 1$,*

$$p_t \leftarrow w_t / \left\| w_t \right\|_1, \qquad \textit{and} \qquad w_{t+1}(i) \leftarrow w_t(i)(1 - \eta v_t(i) + \eta^2 v_t(i)^2). \qquad (6)$$

The following lemma establishes a regret bound for the Variance MW algorithm.

**Lemma 4.2** (Variance MW Lemma). *The Variance MW algorithm satisfies*

$$\sum_{t \in [T]} p_t^\top v_t \leq \min_{i \in [n]} \sum_{t \in [T]} \max\{v_t(i), -1/\eta\} + \frac{\log n}{\eta} + \eta \sum_{t \in [T]} p_t^\top v_t^2.$$

We now state the main theorem. Due to space limitations, we only give here a sketch of the proof.

**Theorem 4.3** (Main). *The SIMBA algorithm above returns an $\varepsilon$-approximate solution to formulation (3) with probability at least $1/2$. It can be implemented to run in time $\tilde{O}(\varepsilon^{-2}(n+d))$.*

*Proof (sketch).* The main idea of the proof is to establish lower and upper bounds on the average objective value $\frac{1}{T} \sum_{t \in [T]} p_t^\top (\mathbf{X} \mathbf{w}_t + \xi_t)$. Then, combining these bounds we are able to relate the value of the output solution $(\bar{\mathbf{w}}, \bar{\xi})$ to the value of the optimum of (3). In the following, we let $(\mathbf{w}^*, \xi^*)$ be the optimal solution of (3) and denote the value of this optimum by $\gamma^*$.

For the lower bound, we consider the primal part of the algorithm. Noting that $\sum_{t \in [T]} p_t^\top \xi_t \geq \sum_{t \in [T]} p_t^\top \xi^*$ (which follows from the PST step) and employing a standard regret guarantee for bounding the regret of the SGD update, we obtain the lower bound (with probability $\geq 1 - O(\frac{1}{n})$):

$$\frac{1}{T} \sum_{t \in [T]} p_t^\top (\mathbf{X} \mathbf{w}_t + \xi_t) \geq \gamma^* - O\left(\sqrt{\tfrac{\log n}{T}}\right).$$

For the upper bound, we examine the dual part of the algorithm. Applying lemma 4.2 for bounding the regret of the MW update, we get the following upper bound (with probability $> \frac{3}{4} - O(\frac{1}{n})$):

$$\frac{1}{T} \sum_{t \in [T]} p_t^\top (\mathbf{X} \mathbf{w}_t + \xi_t) \leq \frac{1}{T} \min_{i \in [n]} \sum_{t \in [T]} [\mathbf{x}_i^\top \mathbf{w}_t + \xi_t(i)] + O\left(\sqrt{\tfrac{\log n}{T}}\right).$$

Relating the two bounds we conclude that $\min_{i \in [n]} [\mathbf{x}_i^\top \bar{\mathbf{w}} + \bar{\xi}(i)] \geq \gamma^* - O(\sqrt{\log(n)/T})$ with probability $\geq \frac{1}{2}$, and using our choice for $T$ the claim follows.

Finally, we note the runtime. The algorithm makes $T = O(\varepsilon^{-2} \log n)$ iterations. In each iteration, the update of the vectors $w_t$ and $p_t$ takes $O(d)$ and $O(n)$ time respectively, while $\xi_t$ can be computed in $O(n)$ time as explained above. The overall runtime is therefore $\tilde{O}(\varepsilon^{-2}(n+d))$. $\qquad \square$

**Incorporating a bias term** We return to the optimization problem (3) presented in section 2, and show how the bias term $b$ can be integrated into our algorithm. Unlike with SGD-based approaches, including the bias term in our framework is straightforward. The only modification required to our algorithm as presented in figure 1 occurs in lines 5 and 9, where the vector $\xi_t$ is referred. For additionally maintaining a bias $b_t$, we change the optimization over $\xi$ in line 5 to a joint optimization over both $\xi$ and $b$:

$$(\xi_t, b_t) \leftarrow \operatorname*{argmax}_{\xi \in \Xi_\nu, \, b \in [-1,1]} p_t^\top (\xi + b \cdot y)$$

and use the computed $b_t$ for the dual update, in line 9: $\tilde{v}_t(i) \leftarrow \mathbf{x}_i(j_t) \left\| \mathbf{w}_t \right\|^2 / \mathbf{w}_t(j_t) + \xi_t(i) + y_i b_t$, while returning the average bias $\bar{b} = \sum_{t \in [T]} b_t / T$ in the output of the algorithm. Notice that we still assume that the labels $y_i$ were subsumed into the instances $\mathbf{x}_i$, as in section 4. The update of $\xi_t$ is thus unchanged and can be carried out as described in section 4. The update of $b_t$, on the other hand, admits a simple, closed-form formula: $b_t = \mathrm{sign}(p_t^\top y)$. Evidently, the running time of each iteration remains $O(n+d)$, as before. The adaptation of the analysis to this case, which involves only a change of constants, is technical and straightforward.

**The sparse case** We conclude the section with a short discussion of the common situation in which the instances are sparse, that is, each instance contains very few non-zero entries. In this case, we can implement algorithm 1 so that each iteration takes $\tilde{O}(\alpha(n+d))$, where $\alpha$ is the overall data sparsity ratio. Implementing the vector updates is straightforward, using a data representation similar to [12]. In order to implement the sampling operations in time $O(\log n)$ and $O(\log d)$, we maintain a tree over the points and coordinates, with internal nodes caching the combined (unnormalized) probability mass of their descendants.

## 5 Runtime Analysis for Learning

In Section 4 we saw how to obtain an $\varepsilon$-approximate solution to the optimization problem (3) in time $\tilde{O}(\varepsilon^{-2}(n+d))$. Combining this with Lemma 2.1, we see that for any Pareto optimal point $\mathbf{w}^*$ of (1) with $\|\mathbf{w}^*\| = B$ and $\hat{R}_{\text{hinge}}(\mathbf{w}^*) = \hat{R}^*$, the runtime required for our method to find a predictor with $\|\mathbf{w}\| \leq 2B$ and $\hat{R}_{\text{hinge}}(\mathbf{w}) \leq \hat{R}^* + \hat{\delta}$ is

$$\tilde{O}\left( B^2(n+d)\left( \frac{\hat{R}^* + \hat{\delta}}{\hat{\delta}} \right)^2 \right). \tag{7}$$

This guarantee is rather different from guarantee for other SVM optimization approaches. E.g. using a stochastic gradient descent (SGD) approach, we could find a predictor with $\|\mathbf{w}\| \leq B$ and $\hat{R}_{\text{hinge}}(\mathbf{w}) \leq \hat{R}^* + \hat{\delta}$ in time $O(B^2 d/\hat{\delta}^2)$. Compared with SGD, we only ensure a constant factor approximation to the norm, and our runtime does depend on the training set size $n$, but the dependence on $\hat{\delta}$ is more favorable. This makes it difficult to compare the guarantees and suggests a different form of comparison is needed. Following [13], instead of comparing the runtime to achieve a certain optimization accuracy on the empirical optimization problem, we analyze the runtime to achieve a desired *generalization* performance.

Recall that our true learning objective is to find a predictor with low generalization error $\mathbf{R}_{\text{err}}(\mathbf{w}) = \Pr_{(\mathbf{x},y)}(y\langle \mathbf{w}, \mathbf{x}\rangle \leq 0)$ where $\mathbf{x}, y$ are distributed according to some unknown source distribution, and the training set is drawn i.i.d. from this distribution. We assume that there exists some (unknown) predictor $\mathbf{w}^*$ that has norm $\|\mathbf{w}^*\| \leq B$ and low expected hinge loss $\mathbf{R}^* = \mathbf{R}_{\text{hinge}}(\mathbf{w}^*) = \mathbf{E}\left[ [1 - y\langle \mathbf{w}^*, \mathbf{x}\rangle]_+ \right]$, and analyze the runtime to find a predictor $\mathbf{w}$ with generalization error $\mathbf{R}_{\text{err}}(\mathbf{w}) \leq \mathbf{R}^* + \delta$.

In order to understand the runtime from this perspective, we must consider the required sample size to obtain generalization to within $\delta$, as well as the required suboptimality for $\|\mathbf{w}\|$ and $\hat{R}_{\text{hinge}}(\mathbf{w})$. The standard SVMs analysis calls for a sample size of $n = O(B^2/\delta^2)$. But since, as we will see, our analysis will be sensitive to the value of $\mathbf{R}^*$, we will consider a more refined generalization guarantee which gives a better rate when $\mathbf{R}^*$ is small relative to $\delta$. Following Theorem 5 of [14] (and recalling that the hinge-loss is an upper bound on margin violations), we have that with high probability over a sample of size $n$, for all predictors $\mathbf{w}$:

$$\mathbf{R}_{\text{err}}(\mathbf{w}) \leq \hat{R}_{\text{hinge}}(\mathbf{w}) + O\left( \frac{\|\mathbf{w}\|^2}{n} + \sqrt{\frac{\|w\|^2 \, \hat{R}_{\text{hinge}}(\mathbf{w})}{n}} \right). \tag{8}$$

This implies that a training set of size

$$n = \tilde{O}\left( \frac{B^2}{\delta} \cdot \frac{\mathbf{R}^* + \delta}{\delta} \right) \tag{9}$$

is enough for generalization to within $\delta$. We will be mostly concerned here with the regime where either $\mathbf{R}^*$ is small and we seek generalization to within $\delta = \Omega(\mathbf{R}^*)$—a typical regime in learning. This is always the case in the realizable setting, where $\mathbf{R}^* = 0$, but includes also the non-realizable setting, as long as the desired estimation error $\delta$ is not much smaller then the unavoidable error $\mathbf{R}^*$. In any case, in such a regime In that case, the second factor in (9) is of order one.

In fact, an online approach [1] can find a predictor with $\mathbf{R}_{\text{err}}(\mathbf{w}) \leq \mathbf{R}^* + \delta$ with a single pass over $n = \tilde{O}(B^2/\delta \cdot (\delta + \mathbf{R}^*)/\delta)$ training points. Since each step takes $O(d)$ time (essentially the time

required to read the training point), the overall runtime is:

$$O\left(\frac{B^2}{\delta}d \cdot \frac{\mathbf{R}^* + \delta}{\delta}\right) \ . \tag{10}$$

Returning to our approach, approximating the norm to within a factor of two is fine, as it only effects the required sample size, and hence the runtime by a constant factor. In particular, in order to ensure $\mathbf{R}_{\mathrm{err}}(\mathbf{w}) \le \mathbf{R}^* + \delta$ it is enough to have $\|\mathbf{w}\| \le 2B$, optimize the empirical hinge loss to within $\hat{\delta} = \delta/2$, and use a sample size as specified in (9) (where we actually consider a radius of $2B$ and require generalization to within $\delta/4$, but this is subsumed in the constant factors). Plugging this into the runtime analysis (7) yields:

**Corollary 5.1.** *For any $B \ge 1$ and $\delta > 0$, with high probability over a training set of size, $n = \tilde{O}(B^2/\delta \cdot (\delta + \mathbf{R}^*)/\delta)$, Algorithm 1 outputs a predictor $\mathbf{w}$ with $\mathbf{R}_{err}(\mathbf{w}) \le \mathbf{R}^* + \delta$ in time*

$$\tilde{O}\left(\left(B^2 d + \frac{B^4}{\delta} \cdot \frac{\delta + \mathbf{R}^*}{\delta}\right) \cdot \left(\frac{\delta + \mathbf{R}^*}{\delta}\right)^2\right)$$

*where $\mathbf{R}^* = \inf_{\|\mathbf{w}^*\| \le B} \mathbf{R}_{hinge}(\mathbf{w}^*)$.*

Let us compare the above runtime to the online runtime (10), focusing on the regime where $\mathbf{R}^*$ is small and $\delta = \Omega(\mathbf{R}^*)$ and so $\frac{\mathbf{R}^* + \delta}{\delta} = O(1)$, and ignoring the logarithmic factors hidden in the $\tilde{O}(\cdot)$ notation in Corollary 5.1. To do so, we will first rewrite the runtime in Corollary 5.1 as:

$$\tilde{O}\left(\frac{B^2}{\delta}d \cdot \frac{\mathbf{R}^* + \delta}{\delta} \cdot (\mathbf{R}^* + \delta) + \frac{B^2}{\delta}B^2 \cdot \left(\frac{\mathbf{R}^* + \delta}{\delta}\right)^3\right) \ . \tag{11}$$

In order to compare the runtimes, we must consider the relative magnitudes of the dimensionality $d$ and the norm $B$. Recall that using a norm-regularized approach, such as SVM, makes sense only when $d \gg B^2$. Otherwise, the low dimensionality would guarantee us good generalization, and we wouldn't gain anything from regularizing the norm. And so, at least when $\frac{\mathbf{R}^* + \delta}{\delta} = O(1)$, the first term in (11) is the dominant term and we should compare it with (10). More generally, we will see an improvement as long as $d \gg B^2(\frac{\mathbf{R}^* + \delta}{\delta})^2$.

Now, the first term in (11) is more directly comparable to the online runtime (10), and is always smaller by a factor of $(\mathbf{R}^* + \delta) \le 1$. This factor, then, is the improvement over the online approach, or more generally, over any approach which considers entire sample vectors (as opposed to individual features). We see, then, that our proposed approach can yield a significant reduction in runtime when the resulting error rate is small. Taking into account the hidden logarithmic factors, we get an improvement as long as $(\mathbf{R}^* + \delta) = O(1/\log(B^2/\delta))$.

Returning to the form of the runtime in Corollary 5.1, we can also understand the runtime as follows: Initially, a runtime of $O(B^2d)$ is required in order for the estimates of $\mathbf{w}$ and $p$ to start being reasonable. However, this runtime does not depend on the desired error (as long as $\delta = \Omega(\mathbf{R}^*)$, including when $\mathbf{R}^* = 0$), and after this initial runtime investment, once $\mathbf{w}$ and $p$ are "reasonable", we can continue decreasing the error toward $\mathbf{R}^*$ with runtime that depends only on the norm, but is *independent of the dimensionality*.

## 6 Experiments

In this section we present preliminary experimental results, that demonstrate situations in which our approach has an advantage over SGD-based methods. To this end, we choose to compare the performance of our algorithm to that of the state-of-the-art Pegasos algorithm [12], a popular SGD variant for solving SVM. The experiments were performed with two standard, large-scale data sets:

- The **news20** data set of [9] that has 1,355,191 features and 19,996 examples. We split the data set into a training set of 8,000 examples and a test set of 11,996 examples.
- The **real vs. simulated** data set of McCallum, with 20,958 features and 72,309 examples. We split the data set into a training set of 20,000 examples and a test set of 52,309 examples.

We implemented the SIMBA algorithm exactly as in Section 4, with a single modification: we used a time-adaptive learning rate $\eta_t = \sqrt{\log(n)/t}$ and a similarly an adaptive SGD step-size (in line 5),

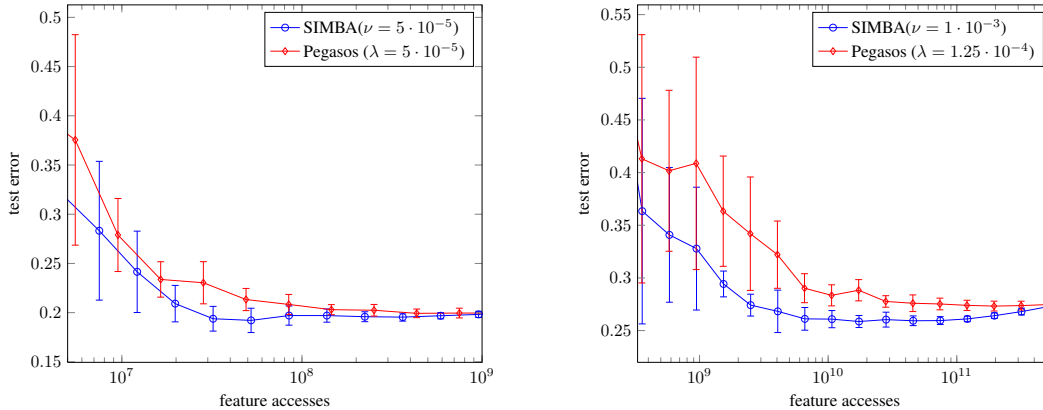

Figure 1: The test error, averaged over 10 repetitions, vs. the number of feature accesses, on the real vs. simulated (left) and news20 (right) data sets. The error bars depict one standard-deviation of the measurements.

instead of leaving them constant. While this version of the algorithm is more convenient to work with, we found that in practice its performance is almost equivalent to that of the original algorithm.

In both experiments, we tuned the tradeoff parameter of each algorithm (i.e., $\nu$ and $\lambda$) so as to obtain the lowest possible error over the test set. Note that our algorithm assumes random access to features (as opposed to instances), thus it is not meaningful to compare the test error as a function of the number of iterations of each algorithm. Instead, and according to our computational model, we compare the test error as a function of the number of *feature accesses* of each algorithm. The results, averaged over 10 repetitions, are presented in figure 1 along with the parameters we used. As can be seen from the graphs, on both data sets our algorithm obtains the same test error as Pegasos achieves at the optimum, using about 100 times less feature accesses.

# 7   Summary

Building on ideas first introduced by [4], we present a stochastic-primal-stochastic-dual approach that solves a non-separable linear SVM optimization problem in sublinear time, and yields a learning method that, in a certain regime, beats SGD and runs in less time than the size of the training set required for learning. We also showed some encouraging preliminary experiments, and we expect further work can yield significant gains, either by improving our method, or by borrowing from the ideas and innovations introduced, including:

- Using importance weighting, and stochastically updating the importance weights in a dual stochastic step.
- Explicitly introducing the slack variables (which are not typically represented in primal SGD approaches). This allows us to differentiate between an accounted-for margin mistakes, and a constraint violation where we did not yet assign enough "slack" and want to focus our attention on. This differs from heuristic importance weighting approaches for stochastic learning, which tend to focus on all samples with a non-zero loss gradient.
- Employing the PST methodology when the standard low-regret tools fail to apply.

We believe that our ideas and framework can also be applied to more complex situations where much computational effort is currently being spent, including highly multiclass and structured SVMs, latent SVMs [6], and situations where features are very expensive to calculate, but can be calculated on-demand. The ideas can also be extended to kernels, either through linearization [11], using an implicit linearization as in [4], or through a representation approach. Beyond SVMs, the framework can apply more broadly, whenever we have a low-regret method for the primal problem, and a sampling procedure for the dual updates. E.g. we expect the approach to be successful for $\ell_1$-regularized problems, and are working on this direction.

**Acknowledgments**   This work was supported in part by the IST Programme of the European Community, under the PASCAL2 Network of Excellence, IST-2007-216886. This publication only reflects the authors' views.

## Footnotes

[1]The Perceptron rule, which amounts to SGD on $\mathbf{R}_{\text{hinge}}(\mathbf{w})$, ignoring correctly classified points [7, 3].

# References

[1] S. Arora, E. Hazan, and S. Kale. The multiplicative weights update method: a meta algorithm and applications. *Manuscript*, 2005.

[2] L. Bottou and O. Bousquet. The tradeoffs of large scale learning. *Advances in neural information processing systems*, 20:161–168, 2008.

[3] N. Cesa-Bianchi, A. Conconi, and C. Gentile. On the generalization ability of on-line learning algorithms. *Information Theory, IEEE Transactions on*, 50(9):2050–2057, 2004.

[4] K.L. Clarkson, E. Hazan, and D.P. Woodruff. Sublinear optimization for machine learning. In *2010 IEEE 51st Annual Symposium on Foundations of Computer Science*, pages 449–457. IEEE, 2010.

[5] K. Deng, C. Bourke, S. Scott, J. Sunderman, and Y. Zheng. Bandit-based algorithms for budgeted learning. In *Data Mining, 2007. ICDM 2007. Seventh IEEE International Conference on*, pages 463–468. IEEE, 2007.

[6] P. Felzenszwalb, D. Mcallester, and D. Ramanan. A discriminatively trained, multiscale, deformable part model. In *In IEEE Conference on Computer Vision and Pattern Recognition (CVPR-2008*, 2008.

[7] C. Gentile. The robustness of the p-norm algorithms. *Machine Learning*, 53(3):265–299, 2003.

[8] A. Kapoor and R. Greiner. Learning and classifying under hard budgets. *Machine Learning: ECML 2005*, pages 170–181, 2005.

[9] S.S. Keerthi and D. DeCoste. A modified finite newton method for fast solution of large scale linear SVMs. *Journal of Machine Learning Research*, 6(1):341, 2006.

[10] S.A. Plotkin, D.B. Shmoys, and É. Tardos. Fast approximation algorithms for fractional packing and covering problems. In *Proceedings of the 32nd annual symposium on Foundations of computer science*, pages 495–504. IEEE Computer Society, 1991.

[11] A. Rahimi and B. Recht. Random features for large-scale kernel machines. *Advances in neural information processing systems*, 20:1177–1184, 2008.

[12] S. Shalev-Shwartz, Y. Singer, and N. Srebro. Pegasos: Primal estimated sub-gradient solver for SVM. In *Proceedings of the 24th international conference on Machine learning*, pages 807–814. ACM, 2007.

[13] S. Shalev-Shwartz and N. Srebro. SVM optimization: inverse dependence on training set size. In *Proceedings of the 25th international conference on Machine learning*, pages 928–935, 2008.

[14] N. Srebro, K. Sridharan, and A. Tewari. Smoothness, low noise and fast rates. In *Advances in Neural Information Processing Systems 23*, pages 2199–2207. 2010.

